# Digital Boltzmann VLSI for constraint satisfaction and learning

Michael Murray[†]    Ming-Tak Leung[†]    Kan Boonyanit[†]

Kong Kritayakirana[†]    James B. Burr[†*]    Gregory J. Wolff[‡]

Takahiro Watanabe[‡]    Edward Schwartz[‡]    David G. Stork[‡†]

Allen M. Peterson[†]

[†]Department of Electrical Engineering
Stanford University
Stanford, CA 94305-4055

[‡]Ricoh California Research Center
2882 Sand Hill Road Suite 115
Menlo Park, CA 94025-7022
and

[*]Sun Microsystems
2550 Garcia Ave., MTV-29, room 203
Mountain View, CA 94043

## Abstract

We built a high-speed, digital mean-field Boltzmann chip and SBus board for general problems in constraint satisfaction and learning. Each chip has 32 neural processors and 4 weight update processors, supporting an arbitrary topology of up to 160 functional neurons. On-chip learning is at a theoretical maximum rate of $3.5 \times 10^8$ connection updates/sec; recall is 12000 patterns/sec for typical conditions. The chip's high speed is due to parallel computation of inner products, limited (but adequate) precision for weights and activations (5 bits), fast clock (125 MHz), and several design insights.

# 1  INTRODUCTION

A vast number of important problems can be cast into a form of constraint satisfaction. A crucial difficulty when solving such problems is the fact that there are local minima in the solution space, and hence simple gradient descent methods rarely suffice. Simulated annealing via the Boltzmann algorithm (*BA*) is attractive because it can avoid local minima better than many other methods (Aarts and Korst, 1989). It is well known that the problem of learning also generally has local minima in weight (parameter) space; a Boltzmann algorithm has been developed for learning which is effective at avoiding local minima (Ackley and Hinton, 1985). The *BA* has not received extensive attention, however, in part because of its slow operation which is due to the annealing stages in which the network is allowed to slowly relax into a state of low error. Consequently there is a great need for fast and efficient special purpose VLSI hardware for implementing the algorithm. Analog Boltzmann chips have been described by Alspector, Jayakumar and Luna (1992) and by Arima et al. (1990); both implement stochastic *BA*. Our *digital* chip is the first to implement the deterministic mean field *BA* algorithm (Hinton, 1989), and although its raw throughput is somewhat lower than the analog chips just mentioned, ours has unique benefits in capacity, ease of interfacing and scalability (Burr, 1991, 1992).

# 2  BOLTZMANN THEORY

The problems of constraint satisfaction and of learning are unified through the Boltzmann learning algorithm. Given a partial pattern and a set of constraints, the *BA* completes the pattern by means of annealing (gradually lowering a computational "temperature" until the lowest energy state is found) — an example of constraint satisfaction. Over a set of training patterns, the learning algorithm modifies the constraints to model the relationships in the data.

## 2.1  CONSTRAINT SATISFACTION

A general constraint satisfaction problem over variables $x_i$ (e.g., neural activations) is to find the set $x_i$ that minimize a global energy function $E = -\frac{1}{2}\sum_{ij} w_{ij}x_i x_j$, where $w_{ij}$ are the (symmetric) connection weights between neurons $i$ and $j$ and represent the problem constraints.

There are two versions of the *BA* approach to minimizing $E$. In one version — the *stochastic BA* — each binary neuron $x_i \in \{-1, 1\}$ is polled randomly, independently and repeatedly, and its state is given a candidate perturbation. The probability of acceptance of this perturbation depends upon the amount of the energy change and the temperature. Early in the annealing schedule (i.e., at high temperature) the probability of acceptance is nearly independent of the change in energy; late in annealing (i.e., at low temperature), candidate changes that lead to lower energy are accepted with higher probability.

In the *deterministic* mean field *BA*, each continuous valued neuron ($-1 \leq x_i \leq 1$) is updated simultaneously and in parallel, its new activation is set to $x_i = f(\sum_j w_{ij}x_j)$, where $f(\cdot)$ is a monotonic non-linearity, typically a sigmoid which corresponds to a stochastic unit at a given temperature (assuming independent

inputs). The inverse slope of the non-linearity is proportional to the temperature; at the end of the anneal the slope is very high and $f(\cdot)$ is effectively a step function. It has been shown that if certain non-restrictive assumptions hold, and if the annealing schedule is sufficiently slow, then the final binary states (at 0 temperature) will be those of minimum $E$ (Hinton, 1989, Peterson and Hartman, 1989).

## 2.2   LEARNING

The problem of Boltzmann *learning* is the following: given a network topology of input and output neurons, interconnected by hidden neurons, and given a set of training patterns (input and desired output), find a set of weights that leads to high probability of a desired output activations for the corresponding input activations. In the Boltzmann algorithm such learning is achieved using two main phases — the *Teacher phase* and the *Student phase* — followed by the actual *Weight update*. During the Teacher phase the network is annealed with the inputs and outputs clamped (held at the values provided by the omniscient teacher). During the anneal of the Student phase, only the inputs are clamped — the outputs are allowed to vary. The weights are updated according to:

$$\Delta w_{ij} = \epsilon(\langle x_i^t x_j^t \rangle - \langle x_i^s x_j^s \rangle) \tag{1}$$

where $\epsilon$ is a learning rate and $\langle x_i^t x_j^t \rangle$ the coactivations of neurons $i$ and $j$ at the end of the Teacher phase and $\langle x_i^s x_j^s \rangle$ in at the end of the Student phase (Ackley and Hinton, 1985). Hinton (1989) has shown that Eq. 1 effectively performs gradient descent on the cross-entropy distance between the probability of a state in the Teacher (clamped) and the Student (free-running) phases.

Recent simulations by Galland (1993) have shown limitations of the deterministic *BA* for learning in networks having hidden units directly connected to other hidden units. While his results do not cast doubt on the deterministic *BA* for constraint satisfaction, they do imply that the deterministic *BA* for learning is most successful in networks of a single hidden layer. Fortunately, with enough hidden units this topology has the expressive power to represent all but the most pathological input-output mappings.

# 3   FUNCTIONAL DESIGN AND CHIP OPERATION

Figure 1 shows the functional block diagram of our chip. The most important units are the *Weight memory, Neural processors, Weight update processors, Sigmoid* and *Rotating Activation Storage* (*RAS*), and their operation are best explained in terms of constraint satisfaction and learning.

## 3.1   CONSTRAINT SATISFACTION

For constraint satisfaction, the weights (constraints) are loaded into the *Weight memory*, the form of the transfer function is loaded into the *Sigmoid Unit*, and the values and duration of the annealing temperatures (the annealing schedule) are loaded into the *Temperature Unit*. Then an input pattern is loaded into a bank of the *RAS* to be annealed. Such an anneal occurs as follows: At an initial high

temperature, the 32 Neural processors compute $x_i = \sum_j w_{ij} x_j$ in parallel for the hidden units. A 4 × multiplexing here permits networks of up to 128 neurons to be annealed, with the remaining 32 neurons used as (non-annealed) inputs. Thus our chip supports networks of up to 160 neurons total. These activations are then stored in the Neural Processor Latch and then passed sequentially to the Sigmoid unit, where they are multiplied by the reciprocal of the instantaneous temperature. This Sigmoid unit employs a lookup table to convert the inputs to neural outputs by means of non-linearity $f(\cdot)$. These outputs are sequentially loaded back into the activation store. The temperature is lowered (according to the annealing schedule), and the new activations are calculated as before, and so on. The final set of activations $x_i$ (i.e., at the lowest temperature) represent the solution.

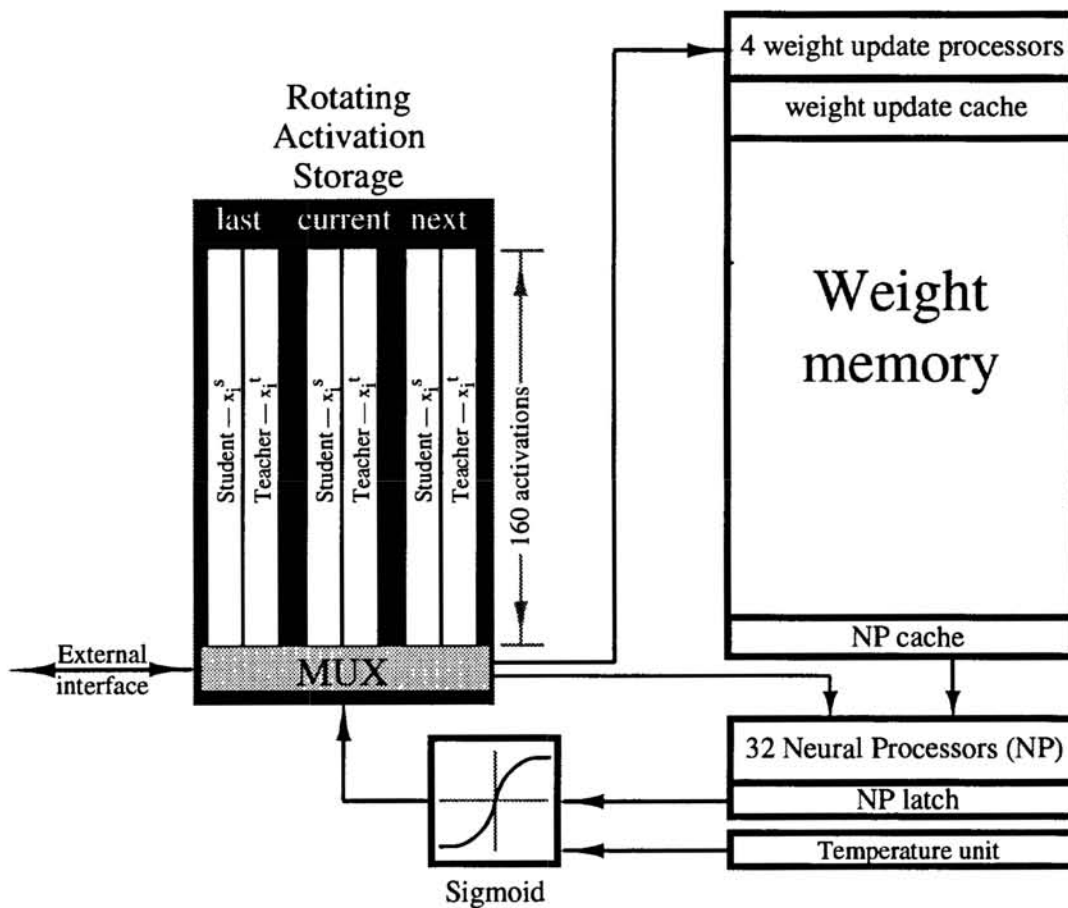

Figure 1: Boltzmann VLSI block diagram. The rotating activation storage (black) consists of three banks, which for learning problems contain the last pattern (already annealed), the current pattern (being annealed) and the next pattern (to be annealed) read onto the chip through the external interface.

## 3.2  LEARNING

When the chip is used for learning, the weight memory is initialized with random weights and the first, second and third training patterns are loaded into the $RAS$. The three-bank $RAS$ is crucial for our chip's speed because it allows a three-fold

concurrency: 1) a *current pattern* of activations is annealed, while 2) the annealed *last pattern* is used to update the weights, while 3) the *next pattern* is being loaded from off-chip. The three banks form a circular buffer, each with a Student and a Teacher activation store.

During the Teacher anneal phase (for the current pattern), activations of the input and output neurons are held at the values given by the teacher, and the values of the hidden units found by annealing (as described in the previous subsection). After the last such annealling step (i.e., at the lowest temperature), the final activations are left in the Teacher activation store — the Teacher phase is then complete. The annealing schedule is then reset to its initial temperature, and the above process is then repeated for the Student phase; here only the input activations are clamped to their values and the outputs are free to vary. At the end of this Student anneal, the final activations are left in the Student activation storage.

In steady state, the *MUX* then rotates the storage banks of the *RAS* such that the next, current, and last banks are now called the current, last, and next, respectively. To update the weights, the activations in the Student and Teacher storage bank for the pattern just annealed (now called the "last" pattern) are sent to the four Weight update processors, along with the weights themselves. The Weight update processors compute the updated weights according to Eq. 1, and write them back to the Weight memory. While such weight update is occuring for the last pattern, the current pattern is annealing and the next pattern is being loaded from off chip.

After the chip has been trained with all of the patterns, it is ready for use in recall. During recall, a test pattern is loaded to the input units of an activation bank (Student side), the machine performs a Student anneal and the final output activations are placed in the Student activation store, then read off the chip to the host computer as the result. In a constraint satisfaction problem, we merely download the weights (constraints) and perform a Student anneal.

## 4 HARDWARE IMPLEMENTATION

Figure 2 shows the chip die. The four main blocks of the Weight memory are at the top, surrounded by 32 Neural processors (above and below this memory), and four Weight update processors (between the memory banks). The three banks of the Rotating Activation Store are at the bottom of the chip. The Sigmoid processor is at the lower left, and instruction cache and external interface at the lower right. Most of the rest of the chip consists of clocking and control circuitry.

### 4.1 VLSI

The chip mixes dynamic and static memory on the same die. The Activation and Temperature memories are static RAM (which needs no refresh circuitry) while the Weight memory is dynamic (for area efficiency). The system clock is distributed to various local clock drivers in order to reduce the global clock capacitance and to selectively disable the clocks in inactive subsystems for reducing power consumption. Each functional block has its own finite state machine control which communicates

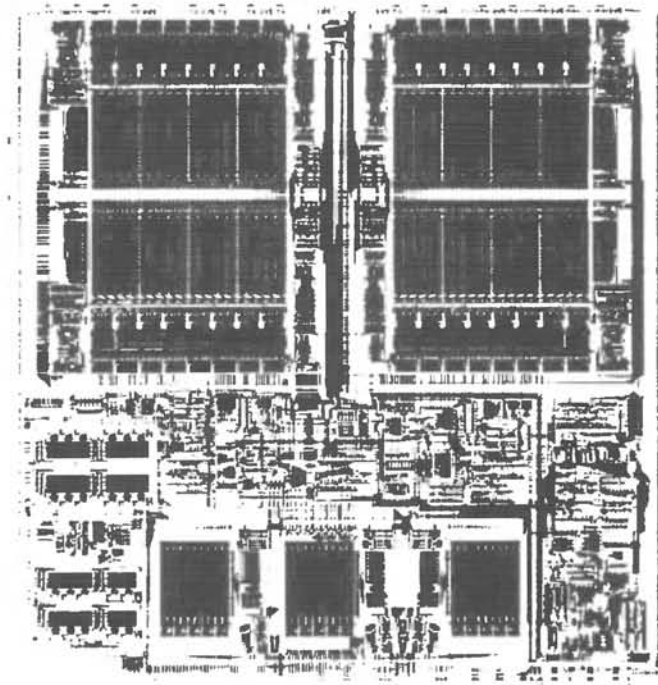

Figure 2: Boltzmann VLSI chip die.

asynchronously. For diagnostic purposes, the State Machines and counters are observable through the External Interface. There is a Single Step mode which has been very useful in verifying sub-system performance. Figure 3 shows the power dissipation throughout a range of frequencies. Note that the power is less than 2 Watts throughout.

Extensive testing of the first silicon revealed two main classes of chip error: electrical and circuit. Most of the electrical problems can be traced to fast edge rates on the DRAM sense-amp equalization control signals, which cause inductive voltage transients on the power supply rails of roughly 1 Volt. This appears to be at least partly responsible for the occasional loss of data in dynamic storage nodes. There also seems to be insufficient latchup protection in the pads, which is aggravated by the on-chip voltage surges. The circuit problems can be traced to having to modify the circuits used in the layout for full chip simulation.

In light of these problems, we have simulated the circuit in great detail in order to explore possible corrective steps. We have modified the design to provide improved electrical isolation, resized drivers and reduced the logic depth in several components. These corrections solve the problems in simulation, and give us confidence that the next fab run will yield a fully working chip.

## 4.2  BOARD AND SBus INTERFACE

An SBus interface board was developed to allow the Boltzmann chip to be used with a SparcStation host. The registers and memory in the chip can be memory mapped so that they are directly accessible to user software. The board can support

Table 1: Boltzmann VLSI chip specifications

| Architecture | n-layer, arbitrary intercorinnections |
|---|---|
| Size | 9.5 mm × 9.8 mm |
| Neurons | 32 processors → 160 virtual |
| Weight memory | 20,480 5-bit weights (on chip) |
| Activation store | 3 banks, 160 teacher & 160 student values in each |
| Technology | $1.2\mu$m CMOS |
| Transistors | 400,000 |
| Pins | 84 |
| Clock | 125 MHz (on chip) |
| I/O rate | $3 \times 10^7$ activations/sec (sustained) |
| Learning rate | $3.5 \times 10^8$ connection updates/sec (on chip) |
| Recall rate | 12000 patterns/sec |
| Power dissipation | $\leq$2 Watts (see Figure 3) |

20-bit transfers to the chip at a sustained rate in excess of 8 Mbytes/second. The board uses reconfigurable Xilinx FPGAs (field-programmable gate arrays) to allow flexibility for testing with and without the chip installed.

## 4.3   SOFTWARE

The chip control program is written in C (roughly 1,500 lines of code) and communicates to the Boltzmann interface card through the virtual memory. The user can read/write to all activation and weight memory locations and all functions of the chip (learning, recall, annealing, etc.) can thus be specified in software.

## 5   CONCLUSIONS AND FUTURE WORK

The chip was designed so that interchip communications could be easily incorporated by means of high-speed parallel busses. The SBus board, interface and software described above will require only minor changes to incorporate a multi-chip module (*MCM*) containing several such chips (for instance 16). There is minimal

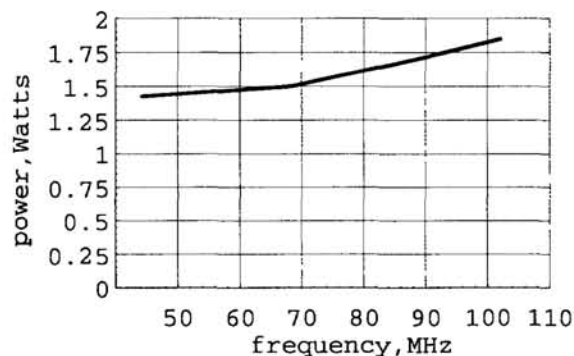

Figure 3: Power dissipation of the chip during full operation at 5 Volts.

interchip communication delay ($< 3\%$ overhead), and thus *MCM* versions of our system promise to be extremely powerful learning systems for large neural network problems (Murray et al., 1992).

## Acknowledgements

Thanks to Martin Boliek and Donald Wynn for assistance in design and construction of the SBus board. Research support by NASA through grant NAGW419 is gratefully acknowledged; VLSI fabrication by MOSIS. Send reprint requests to Dr. Stork: stork@crc.ricoh.com.

## References

E. Aarts & J. Korst. (1989) *Simulated Annealing and Boltzmann Machines: A stochastic approach to combinatorial optimization and neural computing.* New York: Wiley.

D. H. Ackley & G. E. Hinton. (1985) A learning algorithm for Boltzmann machines. *Cognitive Science* **9**, 147-169.

J. Alspector, A. Jayakumar & S. Luna. (1992) Experimental evaluation of learning in a neural microsystem. *Advances in Neural Information Processing Systems-4*, J. E. Moody, S. J. Hanson & R. P. Lippmann (eds.), San Mateo, CA: Morgan Kaufmann, 871-878.

Y. Arima, K. Mashiko, K. Okada, T. Yamada, A. Maeda, H. Kondoh & S. Kayano. (1990) A self-learning neural network chip with 125 neurons and 10K self-organization synapses. In *Symposium on VLSI Circuits*, Solid State Circuits Council Staff, Los Alamitos, CA: IEEE Press, 63-64.

J. B. Burr. (1991) Digital Neural Network Implementations. *Neural Networks: Concepts, Applications, and Implementations, Volume 2*, P. Antognetti & V. Milutinovic (eds.) 237-285, Englewood Cliffs, NJ: Prentice Hall.

J. B. Burr. (1992) Digital Neurochip Design. *Digital Parallel Implementations of Neural Networks.* K. Wojtek Przytula & Viktor K. Prasanna (eds.), Englewood Cliffs, NJ: Prentice Hall.

C. C. Galland. (1993) The limitations of deterministic Boltzmann machine learning. *Network* **4**, 355-379.

G. E. Hinton. (1989) Deterministic Boltzmann learning performs steepest descent in weight-space. *Neural Computation* **1**, 143-150.

C. Peterson & E. Hartman. (1989) Explorations of the mean field theory learning algorithm. *Neural Networks* **2**, 475-494.

M. Murray, J. B. Burr, D. G. Stork, M.-T. Leung, K. Boonyanit, G. J. Wolff & A. M. Peterson. (1992) Deterministic Boltzmann machine VLSI can be scaled using multi-chip modules. *Proc. of the International Conference on Application Specific Array Processors.* Berkeley, CA (August 4-7) Los Alamitos, CA: IEEE Press, 206-217.